# Synchronization of neural networks by mutual learning and its application to cryptography

**Einat Klein**
Department of Physics
Bar-Ilan University
Ramat-Gan, 52900 Israel

**Rachel Mislovaty**
Department of Physics
Bar-Ilan University
Ramat-Gan, 52900 Israel

**Ido Kanter**
Department of Physics
Bar-Ilan University
Ramat-Gan, 52900 Israel

**Andreas Ruttor**
Institut für Theoretische Physik,
Universität Würzbur
Am Hubland 97074 Würzburg, Germany

**Wolfgang Kinzel**
Institut für Theoretische Physik,
Universität Würzbur
Am Hubland 97074 Würzburg, Germany

## Abstract

Two neural networks that are trained on their mutual output synchronize to an identical time dependant weight vector. This novel phenomenon can be used for creation of a secure cryptographic secret-key using a public channel. Several models for this cryptographic system have been suggested, and have been tested for their security under different sophisticated attack strategies. The most promising models are networks that involve chaos synchronization. The synchronization process of mutual learning is described analytically using statistical physics methods.

## 1 Introduction

Neural networks learn from examples. This concept has extensively been investigated using models and methods of statistical mechanics [1, 2]. A "teacher" network is presenting input/output pairs of high dimensional data, and a "student" network is being trained on these data. Training means, that synaptic weights adapt by simple rules to the i/o pairs.

When the networks — teacher as well as student — have $N$ weights, the training process needs of the order of $N$ examples to obtain generalization abilities. This means, that after the training phase the student has achieved some overlap to the teacher, their weight vectors are correlated. As a consequence, the student can classify an input pattern which does not belong to the training set. The average classification error decreases with the number of training examples.

Training can be performed in two different modes: Batch and on-line training. In the first case all examples are stored and used to minimize the total training error. In the second case only one new example is used per time step and then destroyed. Therefore on-line training may be considered as a dynamic process: at each time step the teacher creates a new example which the student uses to change its weights by a tiny amount. In fact, for random input vectors and in the limit $N \to \infty$, learning and generalization can be described by ordinary differential equations for a few order parameters [3].

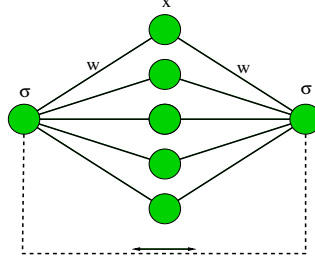

Figure 1: Two perceptrons receive an identical input $\underline{x}$ and learn their mutual output bits $\sigma$.

On-line training is a dynamic process where the examples are generated by a static network - the teacher. The student tries to move towards the teacher. However, the student network itself can generate examples on which it is trained. What happens if two neural networks learn from each other? In the following section an analytic solution is presented [6], which shows a novel phenomenon: synchronization by mutual learning. The biological consequences of this phenomenon are not explored, yet, but we found an interesting application in cryptography: secure generation of a secret key over a public channel.

In the field of cryptography, one is interested in methods to transmit secret messages between two partners A and B. An attacker E who is able to listen to the communication should not be able to recover the secret message.

In 1976, Diffie and Hellmann found a method based on number theory for creating a secret key over a public channel accessible to any attacker[7]. Here we show how neural networks can produce a common secret key by exchanging bits over a public channel and by learning from each other.

## 2 Mutual Learning

We start by presenting the process of mutual learning for a simple network: Two perceptrons receive a common random input vector $\underline{x}$ and change their weights $\underline{w}$ according to their mutual bit $\sigma$, as sketched in Fig. 1. The output bit $\sigma$ of a single perceptron is given by the equation

$$\sigma = \text{sign}(\underline{w} \cdot \underline{x}) \tag{1}$$

$\underline{x}$ is an $N$-dimensional input vector with components which are drawn from a Gaussian with mean 0 and variance 1. $\underline{w}$ is a $N$-dimensional weight vector with continuous components which are normalized,

$$\underline{w} \cdot \underline{w} = 1 \tag{2}$$

The initial state is a random choice of the components $w_i^{A/B}, i = 1, ...N$ for the two weight vectors $\underline{w}^A$ and $\underline{w}^B$. At each training step a common random input vector is presented to the two networks which generate two output bits $\sigma^A$ and $\sigma^B$ according to (1). Now the weight vectors are updated by the perceptron learning rule [3]:

$$
\begin{aligned}
\underline{w}^A(t+1) &= \underline{w}^A(t) + \frac{\eta}{N}\underline{x}\sigma^B \; \Theta(-\sigma^A\sigma^B) \\
\underline{w}^B(t+1) &= \underline{w}^B(t) + \frac{\eta}{N}\underline{x}\sigma^A \; \Theta(-\sigma^A\sigma^B)
\end{aligned}
\tag{3}
$$

$\Theta(x)$ is the step function. Hence, only if the two perceptrons disagree a training step is performed with a learning rate $\eta$. After each step (3), the two weight vectors have to be normalized. In the limit $N \to \infty$, the overlap

$$R(t) = \underline{w}^A(t) \cdot \underline{w}^B(t) \tag{4}$$

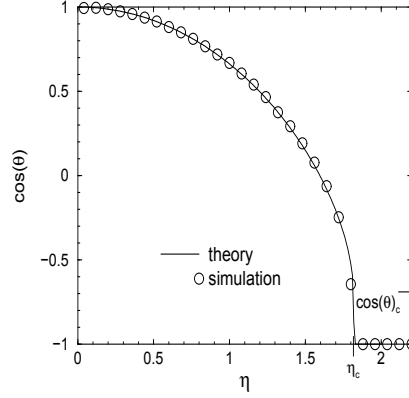

Figure 2: Final overlap $R$ between two perceptrons as a function of learning rate $\eta$. Above a critical rate $\eta_c$ the time dependent networks are synchronized. From Ref. [6]

has been calculated analytically [6]. The number of training steps $t$ is scaled as $\alpha = t/N$, and $R(\alpha)$ follows the equation

$$\frac{dR}{d\alpha} = (R+1)\left(\sqrt{\frac{2}{\pi}}\;\eta(1-R) - \eta^2\frac{\varphi}{\pi}\right) \qquad (5)$$

where $\varphi$ is the angle between the two weight vectors $\underline{w}^A$ and $\underline{w}^B$, i.e. $R = \cos\varphi$. This equation has fixed points $R = 1, R = -1$, and

$$\frac{\eta}{\sqrt{2\pi}} = \frac{1 - \cos\varphi}{\varphi} \qquad (6)$$

Fig. 2 shows the attractive fixed point of (5) as a function of the learning rate $\eta$. For small values of $\eta$ the two networks relax to a state of a mutual agreement, $R \to 1$ for $\eta \to 0$. With increasing learning rate $\eta$ the angle between the two weight vectors increases up to $\varphi = 133°$ for

$$\eta \to \eta_c \cong 1.816 \qquad (7)$$

Above the critical rate $\eta_c$ the networks relax to a state of complete disagreement, $\varphi = 180°, R = -1$. The two weight vectors are antiparallel to each other, $\underline{w}^A = -\underline{w}^B$.

As a consequence, the analytic solution shows, well supported by numerical simulations for $N = 100$, that two neural networks can synchronize to each other by mutual learning. Both networks are trained to the examples generated by their partner and finally obtain an antiparallel alignment. Even after synchronization the networks keep moving, the motion is a kind of random walk on an N-dimensional hypersphere producing a rather complex bit sequence of output bits $\sigma^A = -\sigma^B$ [8].

## 3 Random walk in weight space

We want to apply synchronization of neural networks to cryptography. In the previous section we have seen that the weight vectors of two perceptrons learning from each other can synchronize. The new idea is to use the common weights $\underline{w}^A = -\underline{w}^B$ as a key for encryption [11]. But two issues have to be solved yet: (i) Can an external observer, recording the exchange of bits, calculate the final $\underline{w}^A(t)$ ? The essence of using mutual learning as an encryption tool is the fact that while the parties preform a mutual process in which they

react towards one another, the attacker preforms a learning process, in which the 'teacher' does not react towards him. (ii) Does this phenomenon exist for discrete weights? Since communication is usually based on bit sequences, this is an important practical issue. Both issues are discussed below.

Synchronization occurs for normalized weights, unnormalized ones do not synchronize [6]. Therefore, for discrete weights, we introduce a restriction in the space of possible vectors and limit the components $w_i^{A/B}$ to $2L+1$ different values,

$$w_i^{A/B} \in \{-L, -L+1, ..., L-1, L\} \qquad (8)$$

In order to obtain synchronization to a parallel – instead of an antiparallel – state $\underline{w}^A = \underline{w}^B$, we modify the learning rule (3) to:

$$\underline{w}^A(t+1) = \underline{w}^A(t) - \underline{x}\sigma^A \Theta(\sigma^A \sigma^B) \qquad \underline{w}^B(t+1) = \underline{w}^B(t) - \underline{x}\sigma^B \Theta(\sigma^A \sigma^B) \qquad (9)$$

Now the components of the random input vector $\underline{x}$ are binary $x_i \in \{+1, -1\}$. If the two networks produce an identical output bit $\sigma^A = \sigma^B$, then their weights move one step in the direction of $-x_i\sigma^A$. But the weights should remain in the interval (8), therefore if any component moves out of this interval, $|w_i| = L+1$, it is set back to the boundary $w_i = \pm L$.

Each component of the weight vectors performs a kind of random walk with reflecting boundary. Two corresponding components $w_i^A$ and $w_i^B$ receive the same random number $\pm 1$. After each hit at the boundary the distance $|w_i^A - w_i^B|$ is reduced until it has reached zero. For two perceptrons with a $N$-dimensional weight space we have two ensembles of $N$ random walks on the interval $\{-L, ..., L\}$. We expect that after some characteristic time scale $\tau = \mathcal{O}(L^2)$ the probability of two random walks being in different states decreases as $P(t) \sim P(0)e^{-t/\tau}$. Hence the total synchronization time should be given by $N \cdot P(t) \simeq 1$ which gives $t_{\text{sync}} \sim \tau \ln N$. In fact, our simulations show the synchronization time increases logarithmically with $N$.

## 4 Mutual Learning in the Tree Parity Machine

A single perceptron transmits too much information. An attacker, who knows the set of input/output pairs, can derive the weights of the two partners. On one hand, the information should be hidden so that the attacker does not calculate the weights, but on the other hand enough information should be transmitted so that the two partners can synchronize. We found that multilayer networks with hidden units may be candidates for such a task [11]. More precisely, we consider a Tree Parity Machine(TPM), with three hidden units as shown in Fig. 3.

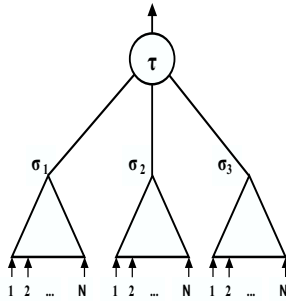

Figure 3: A tree parity machine with $K = 3$

Each hidden unit is a perceptron (1) with discrete weights (8). The output bit $\tau$ of the total network is the product of the three bits of the hidden units

$$\tau^A = \sigma_1^A \sigma_2^A \sigma_3^A \qquad \tau^B = \sigma_1^B \sigma_2^B \sigma_3^B \tag{10}$$

At each training step the two machines $A$ and $B$ receive identical input vectors $\underline{x}_1, \underline{x}_2, \underline{x}_3$. The training algorithm is the following: Only if the two output bits are identical, $\tau^A = \tau^B$, the weights can be changed. In this case, only the hidden unit $\sigma_i$ which is identical to $\tau$ changes its weights using the Hebbian rule

$$\underline{w}_i^A(t+1) = \underline{w}_i^A(t) - \underline{x}_i \tau^A \tag{11}$$

The partner as well as any attacker does not know which one of the K weight vectors is updated. The partners $A$ and $B$ react to their mutual output and move signals $\tau^A$ and $\tau^B$, whereas an attacker can only receive these signals but not influence the partners with its own output bit. This is the essential mechanism which allows synchronization but prohibits learning. Nevertheless, advanced attackers use different heuristics to accelerate their synchronization, as described in the next section.

## 5   Attackers

The following are possible attack strategies, which were suggested by Shamir et al.[12]: *The Genetic Attack*, in which a large population of attackers is trained, and every new time step each attacker is multiplied to cover the $2^{K-1}$ possible internal representations of $\{\sigma_i\}$ for the current output $\tau$. As dynamics proceeds successful attackers stay while the unsuccessful are removed. *The Probabilistic Attack*, in which the attacker tries to follow the probability of every weight element by calculating the distribution of the local field of every input and using the output, which is publicly known. *The Naive Attacker,* in which the attacker imitates one of the parties.

More successful is the *Flipping Attack* strategy, in which the attacker imitates one of the parties, but in steps in which his output disagrees with the imitated party's output, he negates ("flips") the sign of one of his hidden units. The unit most likely to be wrong is the one with the minimal absolute value of the local field, therefore that is the unit which is flipped.

While the synchronization time increases with $L^2$[15], the probability of finding a successful flipping-attacker decreases exponentially with $L$,

$$P \propto e^{-yL}$$

as seen in Figure 4. Therefore, for large $L$ values the system is secure[15]. Every time step, the parties either appraoch each other ("attractive step" or drift apart ("repulsive step"). Close to synchronization the probability for a repulsive step in the mutual learning between $A$ and $B$ scales like $(\epsilon)^2$, while in the dynamic learning between the naive attacker $C$ and $A$ it scales like $\epsilon$, where we define $\epsilon = Prob\left(\sigma_i^C \neq \sigma_i^A\right)$ [18].

It has been shown that among a group of Ising vector students which perform learning, and have an overlap R with the teacher, the best student is the center of mass vector (which was shown to be an Ising vector as well) which has an overlap $R_{cm} \propto \sqrt{R}$, for $R \in [0:1]$[19]. Therefore letting a group of attackers cooperate throughout the process may be to their advantage. The most successful attack strategy, the "Majority Flipping Attacker" uses a group of attackers as a *cooperating group rather than as individuals*. When updating the weights, instead of each attacker being updated according to its own result, all are updated according to the majority's result. This "team-work" approach improves the attacker's performance. When using the majority scheme, the probability for a successful attacker seems to approach a constant value $\sim 0.5$ independent of L.

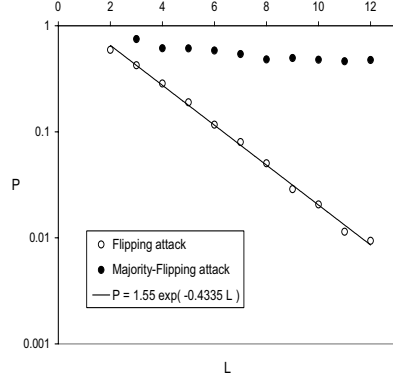

Figure 4: The attacker's success probability $P$ as a function of L, for the flipping attack and the majority-flipping attack, with N=1000, M=100, averaged over 1000 samples. To avoid fluctuations, we define the attacker successful if he found out 98% of the weights

## 6 Analytical description

The semi-analytical description of this process gives us further insight to the synchronization process of mutual and dynamic learning. The study of discrete networks requires different methods of analysis than those used for the continuous case. We found that instead of examining the evolution of $R$ and $Q$, we must examine $(2L+1) \times (2L+1)$ parameters, which describe the mutual learning process. By writing a Markovian process that describes the development of these parameters, one gains an insight into the learning procedure. Thus we define a $(2L+1) \times (2L+1)$ matrix, $\mathbf{F}^\mu$, in which the state of the machines in the time step $\mu$ is represented. The elements of $\mathbf{F}$, are $f_{qr}$, where $q, r = -L, \ldots - 1, 0, 1, \ldots L$. The element $f_{qr}$ represents the fraction of components in a weight vector in which the $A$'s components are equal to $q$ and the matching components in d unit $B$ are equal to $r$. Hence, the overlap between the two units as well as their norms are defined through this matrix,

$$R = \sum_{q,r=-L}^{L} qr f_{qr}, \quad Q^A = \sum_{q=-L}^{L} q^2 f_{qr} Q^B = \sum_{r=-L}^{L} r^2 f_{qr} \qquad (12)$$

The updating of matrix elements is described as follows: for the elements with $q$ and $r$ which are not on the boundary, ($q \neq \pm L$ and $r \neq \pm L$) the update can be written in a simple manner,

$$f_{q,r}^+ = \theta \left( p_\alpha - \epsilon \right) f_{q,r} + \theta \left( \epsilon - p_\alpha \right) \left( \frac{1}{2} f_{q+1,r-1} + \frac{1}{2} f_{q-1,r+1} \right). \qquad (13)$$

Our results indicate that the order parameters are not self-averaged quantities [16]. Several runs with the same $N$, results in different curves for the order parameters as a function of the number of steps, see Figure 5. This explains the non-zero variance of $\rho$ as a results of the fluctuations in the local fields induced by the input even in the thermodynamic limit.

## 7 Combining neural networks and chaos synchronization

Two chaotic system starting from different initial conditions can be synchronized by different kinds of couplings between them. This chaotic synchronization can been used in neural

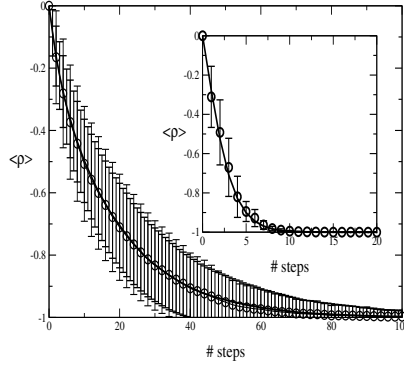

Figure 5: The averaged overlap $\langle \rho \rangle$ and its standard deviation as a function of the number of steps as found from the analytical results (solid line) and simulation results (circles) of mutual learning in TPMs. Inset: analytical results (solid line) and simulation results (circles) results for the perceptron, with $L = 1$ and $N = 10^4$.

cryptography to enhance the cryptographic systems and to improve their security. A model which combines a TPM and logistic maps and is hereby presented, was shown to be more secure than the TPM discussed above. Other models which use mutual synchronization of networks whose dynamics are those of the Lorenz system are now under research and seem very promising.

In the following system we combine neural networks with logistic maps: Both partners A and B use their neural networks as input for the logistic maps which generate the output bits to be learned. By mutually learning these bits, the two neural networks approach each other and produce an identical signal to the chaotic maps which – in turn – synchronize as well, therefore accelerating the synchronization of the neural nets.

Previously, the output bit of each hidden unit was the sign of the local field[11]. Now we combine the PM with chaotic synchronization by feeding the local fields into logistic maps:

$$s_k(t + 1) = \lambda(1 - \beta)s_k(t)(1 - s_k(t)) + \frac{\beta}{2}\tilde{h}_k(t) \tag{14}$$

Here $\tilde{h}$ denotes a transformed local field which is shifted and normalized to fit into the interval $[0, 2]$. For $\beta = 0$ one has the usual quadratic iteration which produces $K$ chaotic series $s_k(t)$ when the parameter $\lambda$ is chosen correspondingly; here we use $\lambda = 3.95$. For $0 < \beta < 1$ the logistic maps are coupled to the fields of the hidden units. It has been shown that such a coupling leads to chaotic synchronization[17]: If two identical maps with different initial conditions are coupled to a *common* external signal they synchronize when the coupling strength is large enough, $\beta > \beta_c$.

The security of key generation increases as the system approaches the critical point of chaotic synchronization. The probability of a successful attack decreases like $\exp(-yL)$ and it is possible that the exponent $y$ diverges as the coupling constant between the neural nets and the chaotic maps is tuned to be critical.

## 8 Conclusions

A new phenomenon has been observed: Synchronization by mutual learning. If the learning rate $\eta$ is large enough, and if the weight vectors keep normalized, then the two networks relax to a parallel orientation. Their weight vectors still move like a random walk on a hypersphere, but each network has complete knowledge about its partner.

It has been shown how this phenomenon can be used for cryptography. The two partners can create a common secret key over a public channel. The fact that the parties are learning mutually, gives them an advantage over the attacker who is learning one-way. In contrast to number theoretical methods the networks are very fast; essentially they are linear filters, the complexity to generate a key of length $N$ scales with $N$ (for sequential update of the weights).

Yet sophisticated attackers which use ensembles of cooperating attackers have a good chance to synchronize. However, advanced algorithms for synchronization, which involve different types of chaotic synchronization seem to be more secure. Such models are subjects of active research, and only the future will tell whether the security of neural network cryptography can compete with number theoretical methods.

## References

[1] J. Hertz, A. Krogh, and R. G. Palmer: *Introduction to the Theory of Neural Computation*, (Addison Wesley, Redwood City, 1991)

[2] A. Engel, and C. Van den Broeck: *Statistical Mechanics of Learning*, (Cambridge University Press, 2001)

[3] M. Biehl and N. Caticha: Statistical Mechanics of On-line Learning and Generalization, *The Handbook of Brain Theory and Neural Networks*, ed. by M. A. Arbib (MIT Press, Berlin 2001)

[4] E. Eisenstein, I. Kanter, D.A. Kessler and W. Kinzel, Phys. Rev. Lett. **74**, 6-9 (1995)

[5] I. Kanter, D.A. Kessler, A. Priel and E. Eisenstein, Phys. Rev. Lett. **75**, 2614-2617 (1995);L. Ein-Dor and I. Kanter, Phys. Rev. **E 57**, 6564 (1998);M. Schröder and W. Kinzel, J. Phys. **A 31**, 9131-9147 (1998); A. Priel and I. Kanter, Europhys. Lett.(2000)

[6] R. Metzler and W. Kinzel and I. Kanter, Phys. Rev. E **62**, 2555 (2000)

[7] D. R. Stinson, *Cryptography: Theory and Practice* (CRC Press 1995)

[8] R. Metzler, W. Kinzel, L. Ein-Dor and I. Kanter, Phys. Rev. **E 63**, 056126 (2001)

[9] M. Rosen-Zvi, I. Kanter and W. Kinzel, cond-mat/0202350 (2002)

[10] R. Urbanczik, private communication

[11] I. Kanter, W. Kinzel and E. Kanter, Europhys. Lett., **57**, 141 (2002).

[12] A.Klimov, A. Mityagin, A. Shamir, ASIACRYPT 2002 : 288-298.

[13] W. Kinzel, R. Metzler and I. Kanter, J. Phys. A. **33** L141 (2000).

[14] W. Kinzel, Contribution to Networks, ed. by H. G. Schuster and S. Bornholdt, to be published by Wiley VCH (2002).

[15] R Mislovaty, Y. Perchenok, I. Kanter and W. Kinzel, Phys. Rev. E **66**, 066102 (2002).

[16] G. Reents and R. Urbanczik, Phys. Rev. Lett., **80**, 5445 (1998).

[17] R. Mislovaty, E. Klein, I. Kanter and W. Kinzel, Phys. Rev. Lett. **91**, 118701 (2003).

[18] M. Rosen-Zvi, E. Klein, I. Kanter and W. Kinzel, Phys. Rev. E **66** 066135 (2002).

[19] M. Copelli, M. Boutin, C. Van Der Broeck and B. Van Rompaey, Europhys. Lett., **46**, 139 (1999).
